# Regularized Policy Iteration

**Amir-massoud Farahmand**[1], **Mohammad Ghavamzadeh**[2], **Csaba Szepesvári**[1], **Shie Mannor**[3] *

[1]Department of Computing Science, University of Alberta, Edmonton, Alberta, Canada
[2]INRIA Lille - Nord Europe, Team SequeL, France
[3]Department of ECE, McGill University, Canada - Department of EE, Technion, Israel

## Abstract

In this paper we consider approximate policy-iteration-based reinforcement learning algorithms. In order to implement a flexible function approximation scheme we propose the use of non-parametric methods with regularization, providing a convenient way to control the complexity of the function approximator. We propose two novel regularized policy iteration algorithms by adding $L^2$-regularization to two widely-used policy evaluation methods: Bellman residual minimization (BRM) and least-squares temporal difference learning (LSTD). We derive efficient implementation for our algorithms when the approximate value-functions belong to a reproducing kernel Hilbert space. We also provide finite-sample performance bounds for our algorithms and show that they are able to achieve optimal rates of convergence under the studied conditions.

## 1 Introduction

A key idea in reinforcement learning (RL) is to learn an action-value function which can then be used to derive a good control policy [15]. When the state space is large or infinite, value-function approximation techniques are necessary, and their quality has a major impact on the quality of the learned policy. Existing techniques include linear function approximation (see, e.g., Chapter 8 of [15]), kernel regression [12], regression tree methods [5], and neural networks (e.g., [13]). The user of these techniques often has to make non-trivial design decisions such as what features to use in the linear function approximator, when to stop growing trees, how many trees to grow, what kernel bandwidth to use, or what neural network architecture to employ. Of course, the best answers to these questions depend on the characteristics of the problem in hand. Hence, ideally, these questions should be answered in an automated way, based on the training data.

A highly desirable requirement for any learning system is to adapt to the actual difficulty of the learning problem. If the problem is easier (than some other problem), the method should deliver better solution(s) with the same amount of data. In the supervised learning literature, such procedures are called *adaptive* [7]. There are many factors that can make a problem easier, such as when only a few of the inputs are relevant, when the input data lies on a low-dimensional submanifold of the input space, when special noise conditions are met, when the expansion of the target function is sparse in a basis, or when the target function is highly smooth. These are called the *regularities* of the problem. An adaptive procedure is built in two steps: 1) designing flexible methods with a few tunable parameters that are able to deliver "optimal" performance for any targeted regularity, provided that their parameters are chosen properly, and 2) tuning the parameters automatically (automatic model-selection).

Smoothness is one of the most important regularities: In regression when the target function has smoothness of order $p$ the optimal rate of convergence of the squared $L^2$-error is $n^{-2p/(2p+d)}$,

where $n$ is the number of data points and $d$ is the dimension of the input space [7]. Hence, the rate of convergence is higher for larger $p$'s. Methods that achieve the optimal rate are more desirable, at least in the limit for large $n$, and seem to perform well in practice. However, only a few methods in the regression literature are known to achieve the optimal rates. In fact, it is known that tree methods with averaging in the leaves, linear methods with piecewise constant basis functions, and kernel estimates do not achieve the optimal rate, while neural networks and regularized least-squares estimators do [7]. An advantage of using a regularized least-squares estimator compared to neural networks is that these estimators do not get stuck in local minima and therefore their training is more reliable.

In this paper we study how to add $L^2$-regularization to value function approximation in RL. The problem setting is to find a good policy in a batch or active learning scenario for infinite-horizon expected total discounted reward Markovian decision problems with continuous state and finite action spaces. We propose two novel policy evaluation algorithms by adding $L^2$-regularization to two widely-used policy evaluation methods in RL: Bellman residual minimization (BRM) [16; 3] and least-squares temporal difference learning (LSTD) [4]. We show how our algorithms can be implemented efficiently when the value-function approximator belongs to a reproducing kernel Hilbert space. We also prove finite-sample performance bounds for our algorithms. In particular, we show that they are able to achieve a rate that is as good as the corresponding regression rate when the value functions belong to a known smoothness class. We further show that this rate of convergence carries through to the performance of a policy found by running policy iteration with our regularized policy evaluation methods. The results indicate that from the point of view of convergence rates RL is not harder than regression estimation, answering an open question of Antos et al. [2]. Due to space limitations, we do not present the proofs of our theorems in the paper; they can be found, along with some empirical results using our algorithms, in [6].

To our best knowledge this is the first work that addresses finite-sample performance of a *regularized* RL algorithm. While regularization in RL has not been thoroughly explored, there has been a few works that used regularization. Xu et al. [17] used sparsification in LSTD. Although sparsification does achieve some form of regularization, to the best of our knowledge the effect of sparsification on generalization error is not well-understood. Note that sparsification is fundamentally different from our approach. In our method the empirical error and the penalties jointly determine the solution, while in sparsification first a subset of points is selected independently of the empirical error, which are then used to obtain a solution. Comparing the efficiency of these methods requires further research, but the two methods can be combined, as was done in our experiments. Jung and Polani [9] explored adding regularization to BRM, but this solution is restricted to deterministic problems. The main contribution of that work was the development of fast incremental algorithms using sparsification techniques. $L^1$ penalties have been considered by [11], who were similarly concerned with incremental implementations and computational efficiency.

## 2 Preliminaries

As we shall work with continuous spaces, we first introduce a few concepts from analysis. This is followed by an introduction to Markovian Decision Processes (MDPs) and the associated concepts and notation.

For a measurable space with domain $\mathcal{S}$, we let $\mathcal{M}(\mathcal{S})$ and $B(\mathcal{S}; L)$ denote the set of probability measures over $\mathcal{S}$ and the space of bounded measurable functions with domain $\mathcal{S}$ and bound $0 < L < \infty$, respectively. For a measure $\nu \in \mathcal{M}(\mathcal{S})$, and a measurable function $f : \mathcal{S} \to \mathbb{R}$, we define the $L^2(\nu)$-norm of $f$, $\|f\|_\nu$, and its empirical counterpart $\|f\|_{\nu,n}$ as follows:

$$\|f\|_\nu^2 = \int |f(s)|^2 \nu(ds) \,, \qquad \|f\|_{\nu,n}^2 \stackrel{\text{def}}{=} \frac{1}{n} \sum_{t=1}^n f^2(s_t) \,, \quad s_t \sim \nu. \qquad (1)$$

If $\{s_t\}$ is ergodic, $\|f\|_{\nu,n}^2$ converges to $\|f\|_\nu^2$ as $n \to \infty$.

A *finite-action discounted MDP* is a tuple $(\mathcal{X}, \mathcal{A}, P, S, \gamma)$, where $\mathcal{X}$ is the state space, $\mathcal{A} = \{a_1, a_2, \ldots, a_M\}$ is the finite set of $M$ actions, $P : \mathcal{X} \times \mathcal{A} \to \mathcal{M}(\mathcal{X})$ is the transition probability kernel with $P(\cdot|x,a)$ defining the next-state distribution upon taking action $a$ in state $x$, $S(\cdot|x,a)$

gives the corresponding distribution of immediate rewards, and $\gamma \in (0, 1)$ is a discount factor. We make the following assumptions on MDP:

**Assumption A1 (MDP Regularity)** The set of states $\mathcal{X}$ is a compact subspace of the $d$-dimensional Euclidean space and the expected immediate rewards $r(x, a) = \int r S(dr|x, a)$ are bounded by $R_{\max}$.

We denote by $\pi : \mathcal{X} \to \mathcal{M}(\mathcal{A})$ a *stationary Markov policy*. A policy is deterministic if it is a mapping from states to actions $\pi : \mathcal{X} \to \mathcal{A}$. The *value* and the *action-value functions* of a policy $\pi$, denoted respectively by $V^{\pi}$ and $Q^{\pi}$, are defined as the expected sum of discounted rewards that are encountered when the policy $\pi$ is executed:

$$V^{\pi}(x) = \mathbb{E}_{\pi}\left[\sum_{t=0}^{\infty} \gamma^t R_t \middle| X_0 = x\right] \qquad , \qquad Q^{\pi}(x, a) = \mathbb{E}_{\pi}\left[\sum_{t=0}^{\infty} \gamma^t R_t \middle| X_0 = x, A_0 = a\right].$$

Here $R_t$ denotes the reward received at time step $t$; $R_t \sim S(\cdot|X_t, A_t)$, $X_t$ evolves according to $X_{t+1} \sim P(\cdot|X_t, A_t)$, and $A_t$ is sampled from the policy $A_t \sim \pi(\cdot|X_t)$. It is easy to see that for any policy $\pi$, the functions $V^{\pi}$ and $Q^{\pi}$ are bounded by $V_{\max} = Q_{\max} = R_{\max}/(1-\gamma)$. The action-value function of a policy is the unique fixed-point of the Bellman operator $T^{\pi} : B(\mathcal{X} \times \mathcal{A}) \to B(\mathcal{X} \times \mathcal{A})$ defined by

$$(T^{\pi}Q)(x, a) = r(x, a) + \gamma \int Q(y, \pi(y)) P(dy|x, a).$$

Given an MDP, the goal is to find a policy that attains the best possible values, $V^*(x) = \sup_{\pi} V^{\pi}(x), \forall x \in \mathcal{X}$. Function $V^*$ is called the *optimal value function*. Similarly the *optimal action-value function* is defined as $Q^*(x, a) = \sup_{\pi} Q^{\pi}(x, a), \forall x \in \mathcal{X}, \forall a \in \mathcal{A}$. We say that a deterministic policy $\pi$ is *greedy* w.r.t. an action-value function $Q$ and write $\pi = \hat{\pi}(\cdot; Q)$, if, $\pi(x) \in \operatorname{argmax}_{a \in \mathcal{A}} Q(x, a), \forall x \in \mathcal{X}, \forall a \in \mathcal{A}$. Greedy policies are important because any greedy policy w.r.t. $Q^*$ is optimal. Hence, knowing $Q^*$ is sufficient for behaving optimally. In this paper we shall deal with a variant of the policy iteration algorithm [8]. In the basic version of policy iteration an optimal policy is found by computing a series of policies, each being greedy w.r.t. the action-value function of the previous one.

Throughout the paper we denote by $\mathcal{F}^M \subset \{f : \mathcal{X} \times \mathcal{A} \to \mathbb{R}\}$ some subset of real-valued functions over the state-action space $\mathcal{X} \times \mathcal{A}$, and use it as the set of admissible functions in the optimization problems of our algorithms. We will treat $f \in \mathcal{F}^M$ as $f \equiv (f_1, \ldots, f_M), f_j(x) = f(x, a_j), j = 1, \ldots, M$. For $\nu \in \mathcal{M}(\mathcal{X})$, we extend $\|\cdot\|_{\nu}$ and $\|\cdot\|_{\nu,n}$ defined in Eq. (1) to $\mathcal{F}^M$ by $\|f\|_{\nu}^2 = \frac{1}{M} \sum_{j=1}^{M} \|f_j\|_{\nu}^2$, and

$$\|f\|_{\nu,n}^2 = \frac{1}{nM} \sum_{t=1}^{n} \sum_{j=1}^{M} \mathbb{I}_{\{A_t = a_t\}} f_j^2(X_t) = \frac{1}{nM} \sum_{t=1}^{n} f^2(X_t, A_t), \tag{2}$$

where $\mathbb{I}_{\{\cdot\}}$ is the indicator function: for an event $E$, $\mathbb{I}_{\{E\}} = 1$ if and only if $E$ holds and $\mathbb{I}_{\{E\}} = 0$, otherwise.

## 3 Approximate Policy Evaluation

The ability to evaluate a given policy is the core requirement to run policy iteration. Loosely speaking, in policy evaluation the goal is to find a "close enough" approximation $V$ (or $Q$) of the value (or action-value) function of a fixed *target policy* $\pi$, $V^{\pi}$ (or $Q^{\pi}$). There are several interpretations to the term "close enough" in this context and it does not necessarily refer to a minimization of some norm. If $Q^{\pi}$ (or noisy estimates of it) is available at a number of points $(X_t, A_t)$, one can form a training set of examples of the form $\{(X_t, A_t), Q_t\}_{1 \le t \le n}$, where $Q_t$ is an estimate of $Q^{\pi}(X_t, A_t)$ and then use a *supervised learning* algorithm to infer a function $Q$ that is meant to approximate $Q^{\pi}$. Unfortunately, in the context of control, the target function, $Q^{\pi}$, is not known in advance and its high quality samples are often very expensive to obtain if this option is available at all. Most often these values have to be inferred from the observed system dynamics, where the observations do not necessarily come from following the target policy $\pi$. This is referred to as the *off-policy learning problem* in the RL literature. The difficulty arising is similar to the problem when training and test distributions differ in supervised learning. Many policy evaluation techniques have been developed in RL. Here we concentrate on the ones that are directly related to our proposed algorithms.

## 3.1 Bellman Residual Minimization

The idea of Bellman residual minimization (BRM) goes back at least to the work of Schweitzer and Seidmann [14]. It was used later in the RL community by Williams and Baird [16] and Baird [3]. The basic idea of BRM comes from writing the fixed-point equation for the Bellman operator in the form $Q^\pi - T^\pi Q^\pi = 0$. When $Q^\pi$ is replaced by some other function $Q$, the left-hand side becomes non-zero. The resulting quantity, $Q - T^\pi Q$, is called the *Bellman residual* of $Q$. If the magnitude of the Bellman residual, $\|Q - T^\pi Q\|$, is small, then $Q$ can be expected to be a good approximation of $Q^\pi$. For an analysis using supremum norms see, e.g., [16]. It seems, however, more natural to use a weighted $L^2$-norm to measure the magnitude of the Bellman residual as it leads to an optimization problem with favorable characteristics and enables an easy connection to regression function estimation [7]. Hence, we define the loss function $L_{BRM}(Q; \pi) = \|Q - T^\pi Q\|_\nu^2$, where $\nu$ is the stationary distribution of states in the input data. Using Eq. (2) with samples $(X_t, A_t)$ and by replacing $(T^\pi Q)(X_t, A_t)$ with its sample-based approximation $(\hat{T}^\pi Q)(X_t, A_t) = R_t + \gamma Q(X_{t+1}, \pi(X_{t+1}))$, the empirical counterpart of $L_{BRM}(Q; \pi)$ can be written as

$$\hat{L}_{BRM}(Q; \pi, n) = \frac{1}{nM} \sum_{t=1}^{n} \left[ Q(X_t, A_t) - \left( R_t + \gamma Q(X_{t+1}, \pi(X_{t+1})) \right) \right]^2. \tag{3}$$

However, as it is well-known (e.g., see [15],[10]), in general, $\hat{L}_{BRM}$ is *not* an unbiased estimate of $L_{BRM}$; $\mathbb{E}\left[ \hat{L}_{BRM}(Q; \pi, n) \right] \neq L_{BRM}(Q; \pi)$. The reason is that stochastic transitions may lead to a non-vanishing variance term in Eq. (3). A common suggestion to deal with this problem is to use uncorrelated or "double" samples in $\hat{L}_{BRM}$. According to this proposal, for each state-action pair in the sample, at least two next states should be generated (e.g., see [15]). This is neither realistic nor sample-efficient unless a generative model of the environment is available or the state transitions are deterministic. Antos et al. [2] recently proposed a de-biasing procedure for this problem. We will refer to it as modified BRM in this paper. The idea is to cancel the unwanted variance by introducing an auxiliary function $h$ and a new loss function $L_{BRM}(Q, h; \pi) = L_{BRM}(Q; \pi) - \|h - T^\pi Q\|_\nu^2$, and approximating the action-value function $Q^\pi$ by solving

$$\hat{Q}_{BRM} = \underset{Q \in \mathcal{F}^M}{\operatorname{argmin}} \ \underset{h \in \mathcal{F}^M}{\sup} \ L_{BRM}(Q, h; \pi), \tag{4}$$

where the supremum comes from the negative sign of $\|h - T^\pi Q\|_\nu^2$. They showed that optimizing the new loss function still makes sense and the empirical version of this loss is unbiased. Solving Eq. (4) requires solving the following nested optimization problems:

$$h_Q^* = \underset{h \in \mathcal{F}^M}{\operatorname{argmin}} \left\| h - \hat{T}^\pi Q \right\|_\nu^2 , \quad \hat{Q}_{BRM} = \underset{Q \in \mathcal{F}^M}{\operatorname{argmin}} \left[ \left\| Q - \hat{T}^\pi Q \right\|_\nu^2 - \left\| h_Q^* - \hat{T}^\pi Q \right\|_\nu^2 \right]. \tag{5}$$

Of course in practice, $T^\pi Q$ is replaced by its sample-based approximation $\hat{T}^\pi Q$.

## 3.2 Least-Squares Temporal Difference Learning

Least-squares temporal difference learning (LSTD) was first proposed by Bradtke and Barto [4], and later was extended to control by Lagoudakis and Parr [10]. They called the resulting algorithm least-squares policy iteration (LSPI), which is an approximate policy iteration algorithm based on LSTD. Unlike BRM that minimizes the distance of $Q$ and $T^\pi Q$, LSTD minimizes the distance of $Q$ and $\Pi T^\pi Q$, the back-projection of the image of $Q$ under the Bellman operator, $T^\pi Q$, onto the space of admissible functions $\mathcal{F}^M$ (see Figure 1). Formally, this means that LSTD minimizes the loss function $L_{LSTD}(Q; \pi) = \|Q - \Pi T^\pi Q\|_\nu^2$. It can also be seen as finding a good approximation for the fixed-point of operator $\Pi T^\pi$. The projection operator $\Pi : B(\mathcal{X} \times \mathcal{A}) \to B(\mathcal{X} \times \mathcal{A})$ is defined by $\Pi f = \operatorname{argmin}_{h \in \mathcal{F}^M} \|h - f\|_\nu^2$. In order to make this minimization problem computationally feasible, it is usually assumed that $\mathcal{F}^M$ is a linear subspace of $B(\mathcal{X} \times \mathcal{A})$. The LSTD solution can therefore be written as the solution of the following nested optimization problems:

$$h_Q^* = \underset{h \in \mathcal{F}^M}{\operatorname{argmin}} \|h - T^\pi Q\|_\nu^2 , \quad \hat{Q}_{LSTD} = \underset{Q \in \mathcal{F}^M}{\operatorname{argmin}} \|Q - h_Q^*\|_\nu^2 , \tag{6}$$

where the first equation finds the projection of $T^\pi Q$ onto $\mathcal{F}^M$, and the second one minimizes the distance of $Q$ and the projection.

Figure 1: This figure shows the loss functions minimized by BRM, modified BRM, and LSTD methods. The function space $\mathcal{F}^M$ is represented by the plane. The Bellman operator, $T^\pi$, maps an action-value function $Q \in \mathcal{F}^M$ to a function $T^\pi Q$. The vector connecting $T^\pi Q$ and its back-projection to $\mathcal{F}^M$, $\Pi T^\pi Q$, is orthogonal to the function space $\mathcal{F}^M$. The BRM loss function is the squared Bellman error, the distance of $Q$ and $T^\pi Q$. In order to obtain the modified BRM loss, the squared distance of $T^\pi Q$ and $\Pi T^\pi Q$ is subtracted from the squared Bellman error. LSTD aims at a function $Q$ that has minimum distance to $\Pi T^\pi Q$.

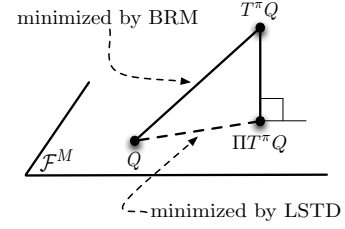

Antos et al. [2] showed that when $\mathcal{F}^M$ is linear, the solution of modified BRM (Eq. 4 or 5) coincides with the LSTD solution (Eq. 6). A quick explanation for this is: the first equations in (5) and (6) are the same, the projected vector $h_Q^* - T^\pi Q$ has to be perpendicular to $\mathcal{F}^M$, as a result $\left\| Q - h_Q^* \right\|^2 = \left\| Q - T^\pi Q \right\|^2 - \left\| h_Q^* - T^\pi Q \right\|^2$ (Pythagorean theorem), and therefore the second equations in (5) and (6) have the same solution.

## 4 Regularized Policy Iteration Algorithms

In this section, we introduce two regularized policy iteration algorithms. These algorithms are instances of the generic policy-iteration method, whose pseudo-code is shown in Table 1. By assumption, the training sample $D_i$ used at the $i$th ($1 \leq i \leq N$) iteration of the algorithm is a finite trajectory $\{(X_t, A_t, R_t)\}_{1 \leq t \leq n}$ generated by a policy $\pi$, thus, $A_t = \pi(X_t)$ and $R_t \sim S(\cdot | X_t, A_t)$. Examples of such policy $\pi$ are $\pi_i$ plus some exploration or some stochastic stationary policy $\pi_b$. The action-value function $Q^{(-1)}$ is used to initialize the first policy. Alternatively, one may start with an arbitrary initial policy. The procedure PEval takes a policy $\pi_i$ (here the greedy policy w.r.t. the current action-value function $Q^{(i-1)}$) along with training sample $D_i$, and returns an approximation to the action-value function of policy $\pi_i$. There are many possibilities to design PEval. In this paper, we propose two approaches, one based on regularized (*modified*) BRM (REG-BRM), and one based on regularized LSTD (REG-LSTD). In REG-BRM, the next iteration is computed by solving the following nested optimization problems:

**FittedPolicyQ**($N$, $Q^{(-1)}$, PEval)
// $N$: number of iterations
// $Q^{(-1)}$: Initial action-value function
// PEval: Fitting procedure
**for** $i = 0$ to $N - 1$ **do**
  $\pi_i(\cdot) \leftarrow \hat{\pi}(\cdot; Q^{(i-1)})$    // the greedy policy w.r.t. $Q^{(i-1)}$ //
  Generate training sample $D_i$
  $Q^{(i)} \leftarrow \text{PEval}(\pi_i, D_i)$
**end for**
**return** $Q^{(N-1)}$ or $\pi_N(\cdot) = \hat{\pi}(\cdot; Q^{(N-1)})$

Table 1: The pseudo-code of policy-iteration algorithm

$$h^*(\cdot; Q) = \underset{h \in \mathcal{F}^M}{\operatorname{argmin}} \left[ \left\| h - \hat{T}^{\pi_i} Q \right\|_n^2 + \lambda_{h,n} J(h) \right] , \quad Q^{(i)} = \underset{Q \in \mathcal{F}^M}{\operatorname{argmin}} \left[ \left\| Q - \hat{T}^{\pi_i} Q \right\|_n^2 - \left\| h^*(\cdot; Q) - \hat{T}^{\pi_i} Q \right\|_n^2 + \lambda_{Q,n} J(Q) \right],$$
(7)

where $(\hat{T}^{\pi_i} Q)(Z_t) = R_t + \gamma Q(Z_t')$ represents the empirical Bellman operator, $Z_t = (X_t, A_t)$ and $Z_t' = (X_{t+1}, \pi_i(X_{t+1}))$ represent state-action pairs, $J(h)$ and $J(Q)$ are penalty functions (e.g., norms), and $\lambda_{h,n}, \lambda_{Q,n} > 0$ are regularization coefficients.

In REG-LSTD, the next iteration is computed by solving the following nested optimization problems:

$$h^*(\cdot; Q) = \underset{h \in \mathcal{F}^M}{\operatorname{argmin}} \left[ \left\| h - \hat{T}^{\pi_i} Q \right\|_n^2 + \lambda_{h,n} J(h) \right] , \quad Q^{(i)} = \underset{Q \in \mathcal{F}^M}{\operatorname{argmin}} \left[ \| Q - h^*(\cdot; Q) \|_n^2 + \lambda_{Q,n} J(Q) \right]. \quad (8)$$

It is important to note that unlike the non-regularized case described in Sections 3.1 and 3.2, REG-BRM and REG-LSTD do not have the same solution. This is because, although the first equations in (7) and (8) are the same, the projected vector $h^*(\cdot; Q) - \hat{T}^{\pi_i} Q$ is not necessarily perpendicular to the admissible function space $\mathcal{F}^M$. This is due to the regularization term $\lambda_{h,n} J(h)$. As a result, the

Pythagorean theorem does not hold: $\|Q - h^*(\cdot; Q)\|^2 \neq \left\|Q - \hat{T}^{\pi_i} Q\right\|^2 - \left\|h^*(\cdot; Q) - \hat{T}^{\pi_i} Q\right\|^2$, and therefore the objective functions of the second equations in (7) and (8) are not equal and they do not have the same solution.

We now present algorithmic solutions for REG-BRM and REG-LSTD problems described above. We can obtain $Q^{(i)}$ by solving the regularization problems of Eqs. (7) and (8) in a reproducing kernel Hilbert space (RKHS) defined by a Mercer kernel $K$. In this case, we let the regularization terms $J(h)$ and $J(Q)$ be the RKHS norms of $h$ and $Q$, $\|h\|_{\mathcal{H}}^2$ and $\|Q\|_{\mathcal{H}}^2$, respectively. Using the Representer theorem, we can then obtain the following closed-form solutions for REG-BRM and REG-LSTD. This is not immediate, because the solutions of these procedures are defined with nested optimization problems.

**Theorem 1.** *The optimizer $Q \in \mathcal{H}$ of Eqs. (7) and (8) can be written as $Q(\cdot) = \sum_{i=1}^{2n} \tilde{\alpha}_i k(\tilde{Z}_i, \cdot)$, where $\tilde{Z}_i = Z_i$ if $i \leq n$ and $\tilde{Z}_i = Z'_{i-n}$, otherwise. The coefficient vector $\tilde{\boldsymbol{\alpha}} = (\tilde{\alpha}_1, \ldots, \tilde{\alpha}_{2n})^\top$ can be obtained by*

$$\text{REG-BRM:} \qquad \tilde{\boldsymbol{\alpha}} = (\boldsymbol{C}\boldsymbol{K}_Q + \lambda_{Q,n}\boldsymbol{I})^{-1}(\boldsymbol{D}^\top + \gamma \boldsymbol{C}_2^\top \boldsymbol{B}^\top \boldsymbol{B})\boldsymbol{r},$$

$$\text{REG-LSTD:} \qquad \tilde{\boldsymbol{\alpha}} = (\boldsymbol{F}^\top \boldsymbol{F} \boldsymbol{K}_Q + \lambda_{Q,n}\boldsymbol{I})^{-1}\boldsymbol{F}^\top \boldsymbol{E}\boldsymbol{r},$$

*where $\boldsymbol{r} = (R_1, \ldots, R_n)^\top$, $\boldsymbol{C} = \boldsymbol{D}^\top \boldsymbol{D} - \gamma^2 (\boldsymbol{B}\boldsymbol{C}_2)^\top (\boldsymbol{B}\boldsymbol{C}_2)$, $\boldsymbol{B} = \boldsymbol{K}_h(\boldsymbol{K}_h + \lambda_{h,n}\boldsymbol{I})^{-1} - \boldsymbol{I}$, $\boldsymbol{D} = \boldsymbol{C}_1 - \gamma\boldsymbol{C}_2$, $\boldsymbol{F} = \boldsymbol{C}_1 - \gamma\boldsymbol{E}\boldsymbol{C}_2$, $\boldsymbol{E} = \boldsymbol{K}_h(\boldsymbol{K}_h + \lambda_{h,n}\boldsymbol{I})^{-1}$, and $\boldsymbol{K}_h \in \mathbb{R}^{n \times n}$, $\boldsymbol{C}_1, \boldsymbol{C}_2 \in \mathbb{R}^{n \times 2n}$, and $\boldsymbol{K}_Q \in \mathbb{R}^{2n \times 2n}$ are defined by $[\boldsymbol{K}_h]_{ij} = k(Z_i, Z_j)$, $[\boldsymbol{C}_1 \boldsymbol{K}_Q]_{ij} = k(Z_i, \tilde{Z}_j)$, $[\boldsymbol{C}_2 \boldsymbol{K}_Q]_{ij} = k(Z'_i, \tilde{Z}_j)$, and $[\boldsymbol{K}_Q]_{ij} = k(\tilde{Z}_i, \tilde{Z}_j)$.*

## 5 Theoretical Analysis of the Algorithms

In this section, we analyze the statistical properties of the policy iteration algorithms based on REG-BRM and REG-LSTD. We provide finite-sample convergence results for the error between $Q^{\pi_N}$, the action-value function of policy $\pi_N$, the policy resulted after $N$ iterations of the algorithms, and $Q^*$, the optimal action-value function. Due to space limitations, we only report assumptions and main results here (Refer to [6] for more details). We make the following assumptions in our analysis, some of which are only technical:

**Assumption A2** (1) At every iteration, samples are generated i.i.d. using a fixed distribution over states $\nu$ and a fixed stochastic policy $\pi_b$, i.e., $\{(Z_t, R_t, Z'_t)\}_{t=1}^n$ are i.i.d. samples, where $Z_t = (X_t, A_t)$, $Z'_t = \left(X'_t, \pi(X'_t)\right)$, $X_t \sim \nu \in \mathcal{M}(\mathcal{X})$, $A_t \sim \pi_b(\cdot|X_t)$, $X'_t \sim P(\cdot|X_t, A_t)$, and $\pi$ is the policy being evaluated. We further assume that $\pi_b$ selects all actions with non-zero probability.

(2) The function space $\mathcal{F}$ used in the optimization problems (7) and (8) is a Sobolev space $\mathbb{W}^k(\mathbb{R}^d)$ with $2k > d$. We denote by $J_k(Q)$ the norm of $Q$ in this Sobolev space.

(3) The selected function space $\mathcal{F}^M$ contains the true action-value function, i.e., $Q^\pi \in \mathcal{F}^M$.

(4) For every function $Q \in \mathcal{F}^M$ with bounded norm $J(Q)$, its image under the Bellman operator, $T^\pi Q$, is in the same space, and we have $J(T^\pi Q) \leq BJ(Q)$, for some positive and finite $B$, which is independent of $Q$.

(5) We assume $\mathcal{F}^M \subset B(\mathcal{X} \times \mathcal{A}; Q_{\max})$, for $Q_{\max} > 0$.

(1) indicates that the training sample should be generated by an i.i.d. process. This assumption is used mainly for simplifying the proofs and can be extended to the case where the training sample is a single trajectory generated by a fixed policy with appropriate mixing conditions as was done in [2]. (2) Using Sobolev space allows us to explicitly show the effect of smoothness $k$ on the convergence rate of our algorithms and to make comparison with the regression learning settings. Note that Sobolev spaces are large: In fact, Sobolev spaces are more flexible than Hölder spaces (a generalization of Lipschitz spaces to higher order smoothness) in that in these spaces the norm measures the *average* smoothness of the functions as opposed to measuring their worst-case smoothness. Thus, functions that are smooth most over the place except for some parts that have a small measure will have small Sobolev-space norms, i.e., they will be looked as "simple", while they would be viewed as "complex" functions in Hölder spaces. Actually, our results extend to other RKHS spaces

that have well-behaved metric entropy capacity, i.e., $\log \mathcal{N}(\varepsilon, \mathcal{F}) \leq A\varepsilon^{-\alpha}$ for some $0 < \alpha < 2$ and some finite positive $A$. In (3), we assume that the considered function space is large enough to include the true action-value function. This is a standard assumption when studying the rate of convergence in supervised learning [7]. (4) constrains the growth rate of the complexity of the norm of $Q$ under Bellman updates. We believe that this is a reasonable assumption that will hold in most practical situations. Finally, (5) is about the uniform boundedness of the functions in the selected function space. If the solutions of our optimization problems are not bounded, they must be truncated, and thus, truncation arguments must be used in the analysis. Truncation does not change the final result, so we do not address it to avoid unnecessary clutter.

We now first derive an upper bound on the policy evaluation error in Theorem 2. We then show how the policy evaluation error propagates through the iterations of policy iteration in Lemma 3. Finally, we state our main result in Theorem 4, which follows directly from the first two results.

**Theorem 2** (Policy Evaluation Error). *Let Assumptions A1 and A2 hold. Choosing $\lambda_{Q,n} = c_1 \left( \frac{\log(n)}{n J_k^2(Q^\pi)} \right)^{\frac{2k}{2k+d}}$ and $\lambda_{h,n} = \Theta(\lambda_{Q,n})$, for any policy $\pi$, the following holds with probability at least $1 - \delta$, for $c_1, c_2, c_3, c_4 > 0$.*

$$\left\| \hat{Q} - T^\pi \hat{Q} \right\|_\nu^2 \leq c_2 \left( J_k^2(Q^\pi) \right)^{\frac{d}{2k+d}} \left( \frac{\log(n)}{n} \right)^{\frac{2k}{2k+d}} + \frac{c_3 \log(n) + c_4 \log(\frac{1}{\delta})}{n}.$$

Theorem 2 shows how the number of samples and the difficulty of the problem as characterized by $J_k^2(Q^\pi)$ influence the policy evaluation error. With a large number of samples, we expect $\|\hat{Q} - T^\pi \hat{Q}\|_\nu^2$ to be small with high probability, where $\pi$ is the policy being evaluated and $\hat{Q}$ is its estimated action-value function using REG-BRM or REG-LSTD.

Let $\hat{Q}^{(i)}$ and $\varepsilon_i = \hat{Q}^{(i)} - T^{\pi_i} \hat{Q}^{(i)}, i = 0, \ldots, N - 1$ denote the estimated action-value function and the Bellman residual at the $i$th iteration of our algorithms. Theorem 2 indicates that at each iteration $i$, the optimization procedure finds a function $\hat{Q}^{(i)}$ such that $\|\varepsilon_i\|_\nu^2$ is small with high probability. Lemma 3, which was stated as Lemma 12 in [2], bounds the final error after $N$ iterations as a function of the intermediate errors. Note that no assumption is made on how the sequence $\hat{Q}^{(i)}$ is generated in this lemma. In Lemma 3 and Theorem 4, $\rho \in \mathcal{M}(\mathcal{X})$ is a measure used to evaluate the performance of the algorithms, and $C_{\rho,\nu}$ and $C_\nu$ are the concentrability coefficients defined in [2].

**Lemma 3** (Error Propagation). *Let $p \geq 1$ be a real and $N$ be a positive integer. Then, for any sequence of functions $\{Q^{(i)}\} \subset B(\mathcal{X} \times \mathcal{A}; Q_{\max}), 0 \leq i < N$, and $\varepsilon_i$ as defined above, the following inequalities hold:*

$$\|Q^* - Q^{\pi_N}\|_{p,\rho} \leq \frac{2\gamma}{(1-\gamma)^2} \left( C_{\rho,\nu}^{1/p} \max_{0 \leq i < N} \|\varepsilon_i\|_{p,\nu} + \gamma^{N/p} R_{\max} \right),$$

$$\|Q^* - Q^{\pi_N}\|_\infty \leq \frac{2\gamma}{(1-\gamma)^2} \left( C_\nu^{1/p} \max_{0 \leq i < N} \|\varepsilon_i\|_{p,\nu} + \gamma^{N/p} R_{\max} \right).$$

**Theorem 4** (Convergence Result). *Let Assumptions A1 and A2 hold, $\lambda_{h,n}$ and $\lambda_{Q,n}$ use the same schedules as in Theorem 2, and the number of samples $n$ be large enough. The error between the optimal action-value function, $Q^*$, and the action-value function of the policy resulted after $N$ iterations of the policy iteration algorithm based on REG-BRM or REG-LSTD, $\hat{Q}^{\pi_N}$, is*

$$\|Q^* - Q^{\pi_N}\|_\rho \leq \frac{2\gamma}{(1-\gamma)^2} \left[ c \times C_{\rho,\nu}^{1/2} \left( \left( \frac{\log(n)}{n} \right)^{\frac{k}{2k+d}} + \left( \frac{\log(\frac{N}{\delta})}{n} \right)^{\frac{1}{2}} \right) + \gamma^{N/2} R_{\max} \right],$$

$$\|Q^* - Q^{\pi_N}\|_\infty \leq \frac{2\gamma}{(1-\gamma)^2} \left[ c \times C_\nu^{1/2} \left( \left( \frac{\log(n)}{n} \right)^{\frac{k}{2k+d}} + \left( \frac{\log(\frac{N}{\delta})}{n} \right)^{\frac{1}{2}} \right) + \gamma^{N/2} R_{\max} \right],$$

*with probability at least $1 - \delta$ for some $c > 0$.*

Theorem 4 shows the effect of number of samples $n$, degree of smoothness $k$, number of iterations $N$, and concentrability coefficients on the quality of the policy induced by the estimated action-value function. Three important observations are: 1) the main term in the rate of convergence is $O(\log(n) n^{-\frac{k}{2k+d}})$, which is an optimal rate for regression up to a logarithmic factor and hence

it is an optimal rate value-function estimation, 2) the effect of smoothness $k$ is evident: for two problems with different degrees of smoothness, learning the smoother one is easier – an intuitive, but previously not rigorously proven result in the RL literature, and 3) increasing the number of iterations $N$ increases the error of the second term, but its effect is only logarithmic.

## 6 Conclusions and Future Work

In this paper we showed how $L^2$-regularization can be added to two widely-used policy evaluation methods in RL: Bellman residual minimization (BRM) and least-squares temporal difference learning (LSTD), and developed two regularized policy evaluation algorithms REG-BRM and REG-LSTD. We then showed how these algorithms can be implemented efficiently when the value-function approximation belongs to a reproducing kernel Hilbert space (RKHS). We also proved finite-sample performance bounds for REG-BRM and REG-LSTD, and the regularized policy iteration algorithms built on top of them. Our theoretical results indicate that our methods are able to achieve the optimal rate of convergence under the studied conditions.

One of the remaining problems is how to find the regularization parameters: $\lambda_{h,n}$ and $\lambda_{Q,n}$. Using cross-validation may lead to a completely self-tuning process. Another issue is the type of regularization. Here we used $L^2$-regularization, however, the idea can be extended naturally to $L^1$-regularization in the style of Lasso, opening up the possibility of procedures that can handle a high number of irrelevant features. Although the i.i.d. sampling assumption is technical, extending our analysis to the case when samples are correlated requires generalizing quite a few results in supervised learning. However, we believe that this can be done without problem following the work of [2]. Extending the results to continuous-action MDPs is another major challenge. Here the interesting question is if it is possible to achieve better rates than the one currently available in the literature, which scales quite unfavorably with the dimension of the action space [1].

## Footnotes

*Csaba Szepesvári is on leave from MTA SZTAKI. This research was funded in part by the National Science and Engineering Research Council (NSERC), iCore and the Alberta Ingenuity Fund. We acknowledge the insightful comments by the reviewers.

## References

[1] A. Antos, R. Munos, and Cs. Szepesvári. Fitted Q-iteration in continuous action-space MDPs. In *Advances in Neural Information Processing Systems 20 (NIPS-2007)*, pages 9–16, 2008.

[2] A. Antos, Cs. Szepesvári, and R. Munos. Learning near-optimal policies with Bellman-residual minimization based fitted policy iteration and a single sample path. *Machine Learning*, 71:89–129, 2008.

[3] L.C. Baird. Residual algorithms: Reinforcement learning with function approximation. In *Proceedings of the Twelfth International Conference on Machine Learning*, pages 30–37, 1995.

[4] S.J. Bradtke and A.G. Barto. Linear least-squares algorithms for temporal difference learning. *Machine Learning*, 22:33–57, 1996.

[5] D. Ernst, P. Geurts, and L. Wehenkel. Tree-based batch mode reinforcement learning. *JMLR*, 6:503–556, 2005.

[6] A. M. Farahmand, M. Ghavamzadeh, Cs. Szepesvári, and S. Mannor. L2-regularized policy iteration. 2009. (under preparation).

[7] L. Györfi, M. Kohler, A. Krzyżak, and H. Walk. *A distribution-free theory of nonparametric regression*. Springer-Verlag, New York, 2002.

[8] R.A. Howard. *Dynamic Programming and Markov Processes*. The MIT Press, Cambridge, MA, 1960.

[9] T. Jung and D. Polani. Least squares SVM for least squares TD learning. In *ECAI*, pages 499–503, 2006.

[10] M. Lagoudakis and R. Parr. Least-squares policy iteration. *JMLR*, 4:1107–1149, 2003.

[11] M. Loth, M. Davy, and P. Preux. Sparse temporal difference learning using LASSO. In *IEEE International Symposium on Approximate Dynamic Programming and Reinforcement Learning*, 2007.

[12] D. Ormoneit and S. Sen. Kernel-based reinforcement learning. *Machine Learning*, 49:161–178, 2002.

[13] M. Riedmiller. Neural fitted Q iteration – first experiences with a data efficient neural reinforcement learning method. In *16th European Conference on Machine Learning*, pages 317–328, 2005.

[14] P. J. Schweitzer and A. Seidmann. Generalized polynomial approximations in Markovian decision processes. *Journal of Mathematical Analysis and Applications*, 110:568–582, 1985.

[15] R.S. Sutton and A.G. Barto. *Reinforcement Learning: An Introduction*. Bradford Book. MIT Press, 1998.

[16] R. J. Williams and L.C. Baird. Tight performance bounds on greedy policies based on imperfect value functions. In *Proceedings of the Tenth Yale Workshop on Adaptive and Learning Systems*, 1994.

[17] X. Xu, D. Hu, and X. Lu. Kernel-based least squares policy iteration for reinforcement learning. *IEEE Trans. on Neural Networks*, 18:973–992, 2007.
